# CoSW: Conditional Sample Weighting for Smoke Segmentation with Label Noise

**Lujian Yao    Haitao Zhao**[*]  **Zhongze Wang    Kaijie Zhao    Jingchao Peng**
School of Information Science and Engineering
East China University of Science and Technology
Shanghai, China
`{lujianyao,zzwang,kjzhao,pjc}@mail.ecust.edu.cn,`
`haitaozhao@ecust.edu.cn`

## Abstract

Smoke segmentation is of great importance in precisely identifying the smoke location, enabling timely fire rescue and gas leak detection. However, due to the visual diversity and blurry edges of the non-grid smoke, noisy labels are almost inevitable in large-scale pixel-level smoke datasets. Noisy labels significantly impact the robustness of the model and may lead to serious accidents. Nevertheless, currently, there are no specific methods for addressing noisy labels in smoke segmentation. Smoke differs from regular objects as its *transparency varies*, causing inconsistent features in the noisy labels. In this paper, we propose a *conditional sample weighting* (CoSW). CoSW utilizes a multi-prototype framework, where prototypes serve as prior information to apply different weighting criteria to the different feature clusters. A novel *regularized within-prototype entropy* (RWE) is introduced to achieve CoSW and stable prototype update. The experiments show that our approach achieves SOTA performance on both real-world and synthetic noisy smoke segmentation datasets.

## 1 Introduction

Smoke segmentation holds significant research value as it enables precise localization of smoke. In wildlife, smoke serves as a vital indicator of fire, and smoke segmentation allows for rapid identification of the source of fire, facilitating prompt rescue [22, 56]. In industrial production, smoke segmentation can identify gas leakage thereby preventing further spread [16]. There have been numerous methods developed for smoke segmentation, ranging from traditional approaches based on color [19, 48] and smoke morphology [11] to deep learning techniques that involve expanding the receptive field [22, 28, 54, 55].

However, to the best of our knowledge, there is no specific work for smoke segmentation with label noise. Noisy labels are almost inevitable in smoke segmentation. Unlike regular objects with clear and concise edges, which are easy to annotate, smoke annotation poses two main challenges: 1) Smoke edges are complex and blurry [53, 55], making it hard to distinguish smoke and background. 2) Smoke is non-rigid [23, 51] and lacks a fixed shape, making it difficult for annotators to become proficient through practice with the same shape.

The noisy labels can have a large impact on the robustness of the model due to their strong memorization power [1, 58]. Given that smoke segmentation is related to safety problems, errors stemming from the instability can lead to significant disasters, resulting in extensive casualties and property losses. Therefore, it is crucial to develop robust training to mitigate the noisy labels in smoke segmentation.

---

[*]Corresponding author.

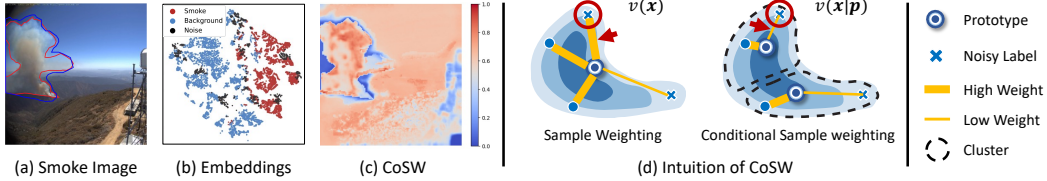

| (a) Smoke Image | (b) Embeddings | (c) CoSW | (d) Intuition of CoSW |

Figure 1: (a) presents the noisy labels in smoke segmentation (blue for noisy labels, and red for clean). (b) shows the pixel features extracted by the encoder of SAM [25]. Noisy labels are variably distributed, with some located along class boundaries and others spread across different regions within the smoke. Additionally, the features demonstrate a polycentric distribution, making them well-suited for prototype-based modeling. In (c), CoSW assigns sample weights that effectively identify regions with noisy labels and reduce their weights. (d) shows the intuition of CoSW.

However, most existing methods primarily focus on noisy labels in classification tasks. Even if a few methods [60, 63] for segmentation, they directly apply classification methods and assume that the noise at the pixel level is also *i.i.d.* (independent and identically distributed).

This assumption is not realistic in smoke segmentation, primarily due to the issue of *variable transparency* in smoke. As depicted in Fig. 1, this problem is ubiquitous in smoke images, mainly caused by the density, size of smoke particles, and lighting conditions, resulting in *inconsistent features* of noisy labels. Noisy labels in the high transparency regions exhibit similarities to the surrounding clean labels, while in the opaque regions, they differ significantly. Treating both types of noisy labels using the same criteria would reduce the accuracy of identifying noisy labels. Moreover, areas with high transparency are more prone to noisy labels and require refined criteria.

In this paper, we propose a *conditional sample weighting* (CoSW), which employs different weighting criteria in different feature clusters to address the problem of feature inconsistency. As illustrated in Fig. 1d, CoSW is built upon a multi-prototype framework and regards prototype as *prior information* for determining the weighting criteria to obtain finer distinctions between samples. Pixel features matching to the same prototype are a feature cluster. Prototype learning is intuitive and concise, with its roots tracing back to the nearest neighbor algorithm [6]. Importantly, by utilizing multiple prototypes, we can establish a polycentric pattern that structures and covers embedding space [4, 62], including both highly transparent and low transparent features.

Under the framework of multi-prototype, CoSW needs to tackle two problems: ① How to determine the sample weight through prototypes. ② How to update prototypes under noisy labels. The most related method is CleanNet [27], which identifies noisy labels based on the similarity between prototypes and samples. However, it only utilizes a single prototype for each class and solely considers individual samples, neglecting the holistic information. As a consequence, it can not dynamically adjust the weighting.

In order to obtain comprehensive information of the samples, a *regularized within-prototype entropy* (RWE) is proposed to address these two problems in a unified manner. Entropy can be used for uncertainty measurement and the maximum entropy principle (MaxEnt) allows for the consideration of the entire probability distribution function (PDF) with minimal empirical risk, rather than focusing solely on individual samples. RWE uses prototypes as anchors to build a separate noisy-level evaluation system for each feature cluster. By maximizing RWE, we can consider the information between all pixels that matched the given prototype and obtain adaptive sample weights (for ①). Furthermore, by calculating the expectation of weighting samples, stable prototypes can be obtained (for ②).

To demonstrate the robustness of our approach, we conduct experiments on both real-world (Smoke-Seg [53] and SMOKE5K [51]) and a synthetic (NS-1K) noisy smoke segmentation dataset. Methods from different fields (semantic segmentation, smoke segmentation, segmentation with label noise, and sample weighting) are compared to demonstrate the superiority of our approach.

Our main contributions can be summarized as follows:

- To the best of our knowledge, we are the first to investigate noisy labels in smoke segmentation.

- We propose a CoSW that serves prototypes as prior information to apply different weighting criteria to the different feature clusters.

- A novel RWE is introduced to implement CoSW, which achieves adaptive sample weighting and stable prototype update in a concise and unified way.

- Our approach achieves SOTA results on real-world and synthetic noisy smoke datasets.

## 2 Related Work

### 2.1 Smoke Segmentation

Traditional smoke segmentation approaches mainly concentrate on extracting color and texture features, including color channel analysis [48] and color enhancement [19]. Additionally, vision-based techniques like morphological operations [11], transmission estimation [30], and region growing [44] have been also widely utilized. Deep-learning-based smoke segmentation methods tackle various challenges related to smoke diversity and ambiguous edges. These approaches encompass multiple aspects, such as 1) fusion of high and low-level features [49, 54, 55], 2) expanding the receptive field [22, 28], and 3) employing coarse-to-fine strategies [56, 59]. 4) uncertainty estimation [51]. However, to the best of our knowledge, there is no work that addresses noisy labels in smoke segmentation.

### 2.2 Sample Importance Weighting

The sample importance weighting (SIW) assigns low weights to potentially mislabeled samples and high weights to potentially confident samples [20, 33, 38]. CleanNet [27] introduces prototype learning for SIW, but it solely relies on the similarity between individual samples and prototypes, neglecting the information from other samples. And it employs only one prototype for each class. There are other techniques like Meta-learning [36, 37, 46], teacher-student architecture [21], iteratively training [50], and transfer learning [29] for SIW. Nevertheless, these methods are all targeted at image-level classification, failing to address the issue of inconsistent features in smoke segmentation with noisy labels.

### 2.3 Prototype Learning

The pioneer of prototype learning is the nearest neighbor algorithm [6]. Building upon this, many nonparametric classification methods have been proposed, including learning vector quantization (LVQ) [26], and neighborhood component analysis (NCA) [12]. In recent years, the concept of prototype learning has also been incorporated into deep learning as it effectively structures and covers the embedding space using a polycentric pattern. Research fields include supervised [31], unsupervised [45], and self-supervised learning [4]. In image segmentation, prototype learning also gains significant attention [9, 43, 62]. However, prototypes are less investigated in the field of noisy labels, and how to update prototypes in a noisy dataset remains an unresolved issue.

### 2.4 Metric Learning

Metric learning maps raw data to an embedding space where similar features are pulled close while dissimilar ones are pushed away. Metric learning and prototype learning can be naturally linked. Some cluster-based methods are using one [24, 32] or multiple [34, 64] learnable prototypes to represent the entire class information. They achieve such mapping through a specific loss function such as contrastive loss [3, 14] and triplet loss [35, 42]. However, employing metric learning without regularization can be detrimental under noisy labels [2, 58].

# 3 Conditional Sample Weighting (CoSW)

## 3.1 Intuition and Overview

Previous research on learning with noisy labels primarily targets classification, with scant attention to segmentation, often presuming noise to be *i.i.d.* However, the noisy labels in smoke segmentation are different due to the variable transparency, resulting in *inconsistent features* within the noisy labels. The objective of CoSW is to apply different weighting criteria to different feature clusters. We achieve CoSW by introducing a regularized within-prototype entropy (RWE). In this section, we first review the concepts of entropy and then introduce a within-prototype entropy (WE) and its regularized form, RWE. Finally, the specific formulation of CoSW is presented and illustrated by a toy demo.

## 3.2 Preliminary: Entropy and MaxEnt

The concept of entropy originates in the realm of thermodynamics, but Shannon has a broader its meaning to the information theory. Entropy can be utilized for measuring uncertainty. For a probability distribution $\Pi = (\pi_1, \pi_2, ..., \pi_N)$ of $N$ random variables $\{x_1, x_2, ..., x_N\}$, Shannon measures the uncertainty for this distribution by $T(\Pi) = -\sum_{i=1}^{N} \pi_i \ln \pi_i$, with the constrain $\sum_{i=1}^{N} \pi_i = 1$.

There is an infinity of probability distributions satisfying the constraint. While maximum entropy theory (MaxEnt) [18] states that, under a given set of constraints, the probability distribution with maximum entropy is the most representative of the current knowledge of a system. Specifically, MaxEnt allows for the consideration of the entire probability distribution with minimal empirical risk, rather than focusing solely on individual samples.

## 3.3 Within-proto & Between-proto Entropy

**Intuition.** Only use probability do not take prototype information into account. Hence, we derive a generalized entropy for prototype information.

**Detail.** Each class $\omega \in \{\omega_1, \omega_2, \cdots, \omega_\Omega\}$ is represented by $K$ prototypes $[\boldsymbol{p}^k]_{k=1}^{K}$ and thus we have $\Omega K$ prototypes in total. We give each pixel feature $\boldsymbol{x}_n^k \in \mathbb{R}^D$ a likelihood value $v_n^k$ and let $\sum_{n=1}^{N^k} v_n^k = N^k$ and $\sum_{k=1}^{\Omega K} \sum_{n=1}^{N^k} v_n^k = \sum_{k=1}^{\Omega K} N^k = N$. Since $\sum_{n=1}^{N^k} v_n^k / N^k = 1$ and $\sum_{k=1}^{\Omega K} \sum_{n=1}^{N^k} v_n^k / N = 1$, we can define the following entropy based on the Shannon entropy:

Total Entropy:

$$T = -\sum_{k=1}^{\Omega K} \sum_{n=1}^{N^k} \frac{v_n^k}{N} \ln \frac{v_n^k}{N}, \tag{1}$$

**Within-prototype Entropy (WE):**

$$T_w = -\sum_{k=1}^{\Omega K} \frac{N^k}{N} \sum_{n=1}^{N^k} \frac{v_n^k}{N^k} \ln(\frac{v_n^k}{N^k}), \tag{2}$$

Between-prototype Entropy:

$$T_b = -\sum_{k=1}^{\Omega K} \frac{N^k}{N} \ln \frac{N^k}{N}, \tag{3}$$

It can be proven that $T = T_w + T_b$. Additionally, we also provide the WE derived from different entropies (*Burg's entropy* and *Kapur's entropy*) as the basis.

## 3.4 Regularized Within-prototype Entropy (RWE)

**Intuition.** Without adding additional constraints, the within-prototype entropy is maximized when all pixel features have the same likelihood value (*i.e.*, $v_n^k = 1$). To tackle the issue of noisy labels, we integrate M-estimation [17] into within-prototype entropy, assigning low weights to noisy labels. The key of original M-estimation is to estimate the mean vector under the noise data. However, in deep learning, obtaining the overall mean of samples becomes challenging due to mini-batch training.

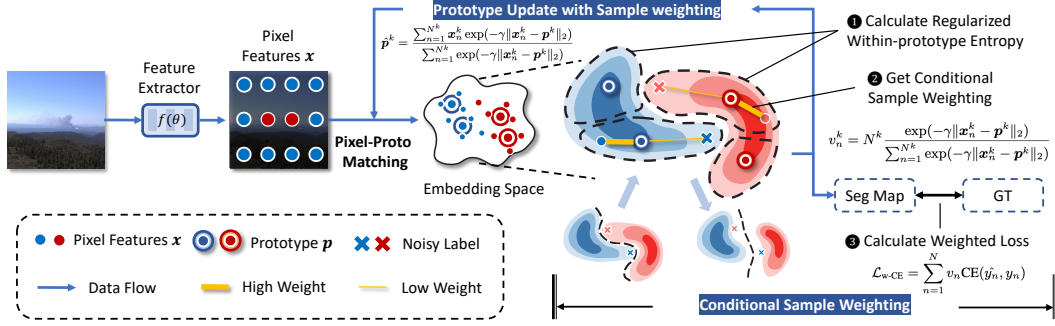

Figure 2: Architecture illustration of CoSW for smoke segmentation during the training process.

Nevertheless, we observe a resemblance between this idea and the concept of prototype learning, where prototypes act as feature representations.

**Detail.** We substituted the mean vector with prototypes to form the constraints:

$$\hat{\boldsymbol{p}}_M(\boldsymbol{x}_n^k) = \arg\min_{\boldsymbol{p}} \sum_n^{N^k} q(\boldsymbol{x}_n^k - \boldsymbol{p}^k) \frac{v_n^k}{N^k}, \tag{4}$$

where $\boldsymbol{p}^k$ represents the corresponding prototype of $\boldsymbol{x}_n^k$, and $q(\cdot)$ denotes the penalty function that measures the influence of the residual error $\boldsymbol{x}_n^k - \boldsymbol{p}^k$. By incorporating this constraint, we can maximize the following objective function in conjunction with the within-prototype entropy:

$$\max_{P,V} J(P,V) = -\sum_{k=1}^{\Omega K} \frac{N^k}{N} \sum_{n=1}^{N^k} \frac{v_n^k}{N^k} \ln(\frac{v_n^k}{N^k}) - \gamma \sum_{k=1}^{\Omega K} \frac{N^k}{N} \sum_{n=1}^{N^k} \frac{v_n^k}{N^k} \|\boldsymbol{x}_n^k - \boldsymbol{p}^k\|_2, \tag{5}$$

with the constraint $\sum_{n=1}^{N^k} v_n^k = N^k$, where $N^k$ denotes the number of pixel features belonging to $\omega$. The penalty function $q(\cdot)$ employs the $L_2$ norm. The $\gamma \in [0,1]$ is a *regularization parameter* which controls the degree of punishment. The function $J(P,V)$ in Eq. 5 is called *regularized within-prototype entropy* (RWE).

**Convergence of the RWE.** The Eq. 5 can be enlarged by:

$$J(P,V) \leq -\sum_{k=1}^{\Omega K} \frac{N^k}{N} \sum_{n=1}^{N^k} \frac{v_n^k}{N^k} \ln(\frac{v_n^k}{N^k}) \leq \frac{1}{N} \sum_{k=1}^{\Omega K} N^k \ln N^k, \tag{6}$$

This proves RWE has an upper bound. According to Cauchy's convergence rule, the RWE is convergent.

### 3.5 Conditional Sample Weighting (CoSW)

The CoSW can be obtained by solving Eq. 5, and the likelihood $v_n^k$ is regarded as the weighting of the sample. We can transform the Eq. 5 to incorporate constraints by Lagrange multipliers and obtain the CoSW $v_n^k$ of each sample and the objective values $\hat{\boldsymbol{p}}^k$ for prototype update:

$$v_n^k = N^k \frac{\exp(-\gamma\|\boldsymbol{x}_n^k - \boldsymbol{p}^k\|_2)}{\sum_{n=1}^{N^k} \exp(-\gamma\|\boldsymbol{x}_n^k - \boldsymbol{p}^k\|_2)}, \tag{7}$$

$$\hat{\boldsymbol{p}}^k = \frac{\sum_{n=1}^{N^k} \boldsymbol{x}_n^k \exp(-\gamma\|\boldsymbol{x}_n^k - \boldsymbol{p}^k\|_2)}{\sum_{n=1}^{N^k} \exp(-\gamma\|\boldsymbol{x}_n^k - \boldsymbol{p}^k\|_2)}, \tag{8}$$

The derivation can be found in the Appendix A.

**Toy Demo.** We provide a toy demo to demonstrate CoSW clearly. Let $X_1 = \{1,1,1,1,\mathbf{5}\}$ and $X_2 = \{-1,-1,-1,-1,\mathbf{0}\}$ be two feature clusters that match to the prototype $p_1 = \{1\}$ and $p_2 = \{-1\}$, respectively. Assume $X_1$ contains an *obvious* noisy label $\{5\}$ and $X_2$ contains an *covert* noisy label $\{0\}$. For normal sample weighting approaches (such as CleanNet), the weights are

calculated by comparing each sample with the center ($\{0.5\}$) of two clusters. The weight of the two noisy labels can be calculated as $v_{(5)} = 0.0282$ and $v_{(0)} = 1.5409$, respectively. This method assigns an excessively high weight to the noisy label $\{0\}$. When we use CoSW, which involves using two prototypes to weigh the importance of each feature separately, we can calculate the $v_{(5)}^{p_1} = 0.0228$ and $v_{(0)}^{p_2} = 0.4211$ through Eq. 7. The noticeable decrease in the weight of the noisy label $\{0\}$. For *prototype updating*, in the case of $X_1$, if CoSW is not applied and only the mean is used for updating, the new prototype $p_1^{'} = \{1.8\}$, which differs from the original value of $\{1\}$. When CoSW is applied and the noisy label $\{5\}$ is given a weight, the new prototype is $p_1^{'} = \{1.0067\}$ (*cf.* Eq. 8), indicating minimal impact on its origin value.

## 4 CoSW for Smoke Segmentation

In this section, we introduce how to apply CoSW to smoke segmentation, including pixel-proto matching, loss design, prototype updating, and regularized scatter metric learning. The entire process is illustrated in Fig. 2.

### 4.1 Pixel-Prototype Matching

**Intuition.** For multi-prototype methods, a crucial task is to match pixels to prototypes. The simplest approach is adopting the nearest neighbors principle. However, this can lead to a large number of pixels being assigned to the same prototype, while the remaining prototypes are left without any matching pixels. Therefore, we apply additional constraints to the original optimization problem to prevent the occurrence of trivial solutions. This part refers to the matching process in ProtoSeg [62] and SwAV [4].

**Detail.** The goal is to match the pixel features $\boldsymbol{X}^{\omega}$ to one of the prototypes in class $\omega$. The matching strategy is denoted as $\boldsymbol{M}^{\omega} \in \mathbb{R}^{K \times N^{\omega}}$. The $\boldsymbol{M}^{\omega}$ is optimized by minimizing the overall distance between each pixel feature (*i.e.*, $\boldsymbol{X}^{\omega} \in \mathbb{R}^{D \times N^{\omega}} = [\boldsymbol{x}_n]_{n=1}^{N}$) and its matched prototype (*i.e.*, $\boldsymbol{P}^{\omega} \in \mathbb{R}^{D \times K} = [\boldsymbol{p}_k^{\omega}]_{k=1}^{K}$):

$$
\min_{\boldsymbol{M}^{\omega}} \mathrm{Tr}(\boldsymbol{M}^{\omega\top}\boldsymbol{C}^{\omega}),
$$
$$
s.t. \quad \boldsymbol{M}^{\omega} \in \{0,1\}^{K \times N^{\omega}}, \ \boldsymbol{M}^{\omega\top}\mathbf{1}^K = \mathbf{1}^{N^{\omega}}, \boldsymbol{M}^{\omega}\mathbf{1}^{N^{\omega}} = \frac{N^{\omega}}{K}\mathbf{1}^K, \tag{9}
$$

where $\mathbf{1}^K$ and $\mathbf{1}^{N^{\omega}}$ denote all-one vectors. $\boldsymbol{M}^{\omega\top}\mathbf{1}^K = \mathbf{1}^{N^{\omega}}$ is a unique matching constrain which guarantees each pixel feature is assigned to only one prototype. $\boldsymbol{M}^{\omega}\mathbf{1}^{N^{\omega}} = \frac{N^{\omega}}{K}\mathbf{1}^K$ is an equipartition constraint [4] which avoids the trivial solution in which all pixels are assigned to a single prototype. And $\boldsymbol{C}^{\omega} \in \mathbb{R}^{K \times N^{\omega}}$ represents the cost matrix that measures the distance between pixel features and prototypes.

Eq. 9 is a typical transportation problem [39] which can be easily calculated by the iterative Sinkhorn-Knopp algorithm [7]. Specific derivation and implementation can be found in the Appendix C.

### 4.2 Sample Weighting and Prototype Update

In the pipeline of smoke segmentation, CoSW is employed in the loss function. We incorporate CoSW into the basic cross-entropy loss:

$$
\mathcal{L}_{\text{w-CE}} = \sum_{n=1}^{N} v_n \mathrm{CE}(\hat{y}_n, y_n), \tag{10}
$$

where $v_n$ refers to Eq. 7. $\hat{y}_n$ and $y_n$ respectively represent the predicted value and ground truth for each pixel in a mini-batch. $N$ represents the number of pixels in a mini-batch.

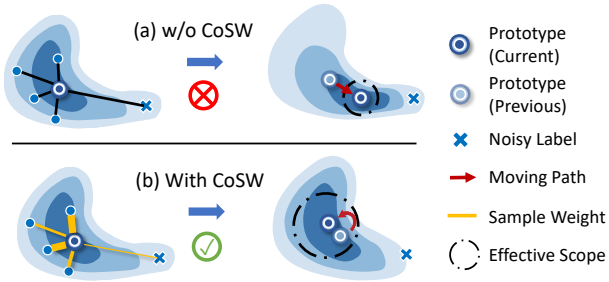

Figure 3: Regularized prototype update.

As for prototype update, it is challenging to ensure stability under noisy labels, as shown in Fig. 3. Our objective for prototype updating is $\hat{\boldsymbol{p}}^k$ (*cf.* Eq. 8) with CoSW. To ensure stable training, we also

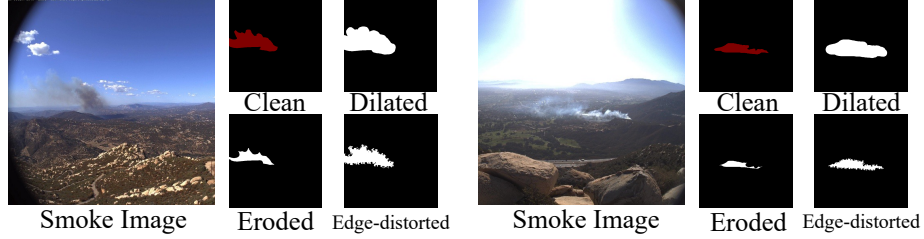

Figure 4: Examples of the clean and three corrupted masks in NS-1K.

incorporate the update in a momentum way [15]:

$$\hat{\boldsymbol{p}}^k \leftarrow \mu \boldsymbol{p}^k + (1-\mu)\hat{\boldsymbol{p}}^k; k = 1, 2, \cdots, \Omega K, \tag{11}$$

where $\mu \in [0, 1]$ is a momentum hyper-parameter.

### 4.3 Regularized Scatter Metric Learning (RSML)

**Intuition.** In prototype framework, metric learning is a widely used and powerful tool [62] that brings similar features closer and pushes dissimilar ones apart, enabling the acquisition of a discriminative embedding space. However, due to the presence of noisy labels, metric learning may lead to overfitting. Therefore, we further incorporate the weighting into the *scatter matrix* to ensure the effectiveness of metric learning. The scatter matrix can capture the dispersion information of the samples [61].

**Detail.** The way we implemented it is by incorporating CoSW into the Within-prototype Scatter Matrix (WSM). The WSM serves as a representation of the dispersion of samples in relation to their corresponding prototypes, allowing for the assessment of the compactness within each prototype. We integrate CoSW $v_n^k$ (*cf.* Eq. 7) into WSM for regularization:

$$\boldsymbol{S}_w = \sum_{k=1}^{\Omega K} \sum_{n=1}^{N^k} \frac{v_n^k}{N} (\boldsymbol{x}_n^k - \boldsymbol{p}^k)(\boldsymbol{x}_n^k - \boldsymbol{p}^k)^T, \tag{12}$$

For the Between-prototype Scatter Matrix (BSM), we do not apply weighting. The BSM can be employed to quantify the separability between prototypes and features. We employ the non-parametric strategy in which the scatter matrix is calculated between each pixel feature and its nearest neighbor prototype. The BSM is defined as follows:

$$\boldsymbol{S}_b = \frac{1}{N} \sum_{n=1}^{N} (\boldsymbol{x}_n^{\omega_0} - \boldsymbol{p}_{NN}(\boldsymbol{x}_n))(\boldsymbol{x}_n^\omega - \boldsymbol{p}_{NN}(\boldsymbol{x}_n^\omega))^T, \tag{13}$$

where $\boldsymbol{p}_{NN}(\boldsymbol{x}_n)$ is the nearest neighbor prototype of $\boldsymbol{x}_n$.

Finally, we employ the ratio-trace form [41] to transform the scatter matrices into a numeric value: $\mathcal{L}_S = \frac{1}{\text{Tr}(\boldsymbol{S}_w'^{-1}\boldsymbol{S}_b)}$, where $\boldsymbol{S}_w' = \boldsymbol{S}_w + \varepsilon\boldsymbol{I}$ and the $\varepsilon\boldsymbol{I}$ is added to guarantee the reversibility of $\boldsymbol{S}_w$ and we set $\varepsilon = 10^{-5}$. DeepLDA [10] has demonstrated that this trace form can be optimized using gradient descent. Our final objective function is a combination of weighted cross-entropy loss $\mathcal{L}_{\text{w-CE}}$ (*cf.* Eq. 10) and scatter loss $\mathcal{L}_S$: $\mathcal{L} = \mathcal{L}_{\text{w-CE}} + \alpha\mathcal{L}_S$, where $\alpha$ is a weight hyperparameter.

## 5 Experiments

### 5.1 Experimental Setup

**Real-world Noise Setting.** Due to the visual diversity and blurry edges of smoke, label noise in the large-scale real-world smoke datasets (SmokeSeg [53] and SMOKE5K [51]) is ubiquitous. Hence, we conduct experiments on both datasets as real-world noise evaluation. To accurately measure the robustness of the model, a clean validation set is essential. Therefore, we carefully re-annotate the validation set of both datasets. The distinction between SmokeSeg and Smoke5K resides in the

Table 1: Quantitive comparison on the real-world noisy dataset SmokeSeg. We compare methods from various domains, including semantic segmentation (∘), smoke segmentation (†), segmentation with label noise (§), and sample weighting (∗). The best: **bold**, the second: underline.

| | Methods | Backbone | Total | | Small | | Medium | | Large | |
|---|---|---|---|---|---|---|---|---|---|---|
| | | | $F_1$ | $mIoU$ | $F_1$ | $mIoU$ | $F_1$ | $mIoU$ | $F_1$ | $mIoU$ |
| *Real-time* | AFFormer∘ [8] | AFFormer-B | 58.41 | 47.89 | 45.98 | 34.10 | 64.52 | 53.20 | 66.96 | 54.15 |
| | SeaFormer∘ [40] | SeaFormer-B | 57.58 | 44.70 | 41.33 | 30.35 | 66.02 | 53.10 | 67.16 | 53.21 |
| | SegFormer∘ [47] | MiT-B0 | 60.90 | 48.53 | 48.31 | **35.69** | **68.86** | 53.34 | 68.53 | 55.80 |
| | SC§ [52] | MiT-B0 | 59.78 | 47.70 | 45.76 | 33.15 | 68.37 | **55.68** | 66.13 | 53.96 |
| | CleanNet∗ [27] | MiT-B0 | 61.63 | 47.00 | 47.16 | 33.57 | 66.23 | 50.91 | **70.96** | **56.22** |
| | Ours | MiT-B0 | **62.98** | **48.62** | **49.85** | 36.64 | 68.81 | 53.87 | 70.79 | 55.85 |
| *Normal* | DeeplabV3+∘ [5] | ResNet-50 | 65.92 | 53.50 | 54.03 | 41.07 | 71.82 | 58.73 | 71.87 | 58.95 |
| | OCRNet∘ [57] | HRNet-48 | 64.93 | 52.45 | 52.04 | 39.47 | 71.04 | 57.47 | 70.51 | 58.19 |
| | SegNeXt∘ [13] | MSCAN-L | 66.71 | 52.37 | 58.05 | 44.16 | 70.41 | 55.77 | 72.97 | 58.42 |
| | Trans-BVM† [51] | ResNet-50 | 67.15 | 53.11 | 59.02 | 44.62 | 71.50 | 56.99 | 73.36 | 58.97 |
| | Ours | ResNet-50 | 68.49 | 54.09 | 61.27 | 46.28 | 72.31 | 58.08 | 74.78 | 60.81 |
| | SegFormer∘ [47] | MiT-B3 | 67.70 | 53.37 | 57.67 | 45.21 | 73.87 | 60.49 | 72.06 | 58.52 |
| | Trans-BVM† [51] | MiT-B3 | 67.68 | 53.09 | 60.85 | 45.87 | 71.73 | 57.35 | 73.51 | 59.10 |
| | SC§ [52] | MiT-B3 | 69.55 | 55.04 | 62.26 | 48.47 | 71.41 | 57.06 | 72.91 | 58.23 |
| | CleanNet∗ [27] | MiT-B3 | 70.17 | 56.94 | 61.98 | 48.05 | 73.06 | 59.07 | 74.57 | 60.93 |
| | Ours | MiT-B3 | **72.32** | **59.25** | **64.62** | **50.86** | **74.37** | **61.14** | **75.52** | **62.30** |

Table 2: Comparison on the real-world noisy dataset SMOKE5K and synthetic noisy dataset NS-1K. Notation: semantic segmentation (∘), smoke segmentation (†), segmentation with label noise (§), sample weighting (∗). The best: **bold**, the second: underline.

(a) Comparison on SMOKE5K.

| Methods | $F_\beta$ | $mIoU$ |
|---|---|---|
| OCRNet∘ [57] | 72.51 | 63.00 |
| DeeplabV3+∘ [5] | 73.83 | 64.08 |
| SegNeXt∘ [13] | 76.44 | 67.08 |
| Trans-BVM† [51] | 76.23 | 67.55 |
| Ours | 77.02 | 67.58 |
| SegFormer∘ [47] | 78.68 | 68.29 |
| Trans-BVM† [51] | 78.91 | 68.97 |
| SC§ [52] | 79.33 | 69.40 |
| CleanNet∗ [27] | 80.37 | 70.23 |
| Ours | **81.71** | **71.24** |

(b) Comparison on the synthetic noisy dataset NS-1K.

| | | Trans-BVM† [51] | | SC§ [52] | | CleanNet∗ [27] | | Ours | |
|---|---|---|---|---|---|---|---|---|---|
| Noise Ratio | Noise Intensity | $F_1$ | $mIoU$ | $F_1$ | $mIoU$ | $F_1$ | $mIoU$ | $F_1$ | $mIoU$ |
| 0% | - | **52.79** | **39.12** | 51.94 | 37.84 | 52.02 | 38.24 | 52.59 | 38.32 |
| 20% | Low | **51.91** | **38.16** | 51.81 | 37.94 | 51.21 | 37.24 | 51.77 | 37.58 |
| | High | 48.02 | 34.52 | **51.14** | **37.15** | 50.34 | 36.31 | 50.99 | 36.80 |
| 40% | Low | 45.40 | 32.30 | 49.69 | 35.89 | 49.09 | 35.53 | **50.18** | **36.69** |
| | High | 41.09 | 28.15 | 45.31 | 32.16 | 46.08 | 33.15 | **48.34** | **34.73** |
| 60% | Low | 42.73 | 29.01 | 44.12 | 30.87 | 46.33 | 33.96 | **48.38** | **35.34** |
| | High | 40.57 | 27.50 | 41.77 | 28.44 | 43.69 | 29.87 | **45.86** | **32.08** |
| 80% | Low | 39.28 | 26.42 | 40.52 | 27.42 | 42.96 | 29.37 | **44.40** | **31.24** |
| | High | 37.38 | 25.58 | 39.09 | 25.88 | 40.27 | 27.21 | **42.37** | **29.89** |

unique smoke samples in SmokeSeg, which display high and variable transparency. Consequently, SmokeSeg offers a better assessment of the robustness of the model.

**Synthetic Noise Setting (NS-1K).** To further investigate the robustness of the model to noise, we also create a synthetic noise smoke segmentation dataset called NS-1K. We select 1,000 images from SmokeSeg and carefully re-annotate them to obtain clean labels. Among them, 700 images are used for training, and 300 images are used for validation. Then, we add noise to this dataset in different forms, including eroded, dilated, and edge-distorted noise, as shown in Fig. 4. It is noted that edge-distorted is implemented by randomly dilating or eroding the pixels on the boundary with a circle. In the experiment, we set two noise parameters: one is the ratio of noise data (20%, 40%, 60%, 80%), and the other is the intensity of the noise data (high and low). The intensity of the noise is determined by adjusting the degree of pixel displacement for three noise types.

**Implementation Details.** We implement our method on MMSegmentation. Standard color jittering, random cropping, and random flipping are adopted for data augmentation during the training stage. All experimental backbones are pre-trained on ImageNet-1K. We utilize the *AdamW* optimizer, with learning rate starting at 6e-5 and scheduled according to the polynomial annealing policy. For the SmokeSeg, we crop images to a size of $512 \times 512$ for training, while for SMOKE5K, we follow the previous methods and resize the images to $480 \times 480$. For validation, we use the whole inference method, and for simplicity, we do not use any data augmentation during the validation.

**Evaluation Metric.** Following previous literature, we employ $F_1$ and $mIoU$ for evaluation.

Table 3: Ablation study of CoSW and metric learning.

(a) Ablation study of CoSW (dataset: SmokeSeg).

|  | Proto | Sample Weight | Proto Weight | $F_1$ | $mIoU$ |
|---|---|---|---|---|---|
| bl |  |  |  | 67.70 | 53.37 |
| (1) | ✓ |  |  | 68.17 ↑ 0.47 | 55.23 ↑ 1.86 |
| (2) | ✓ | ✓ |  | 70.04 ↑ 2.34 | 56.40 ↑ 3.03 |
| (3) | ✓ |  | ✓ | 69.38 ↑ 1.68 | 55.72 ↑ 2.35 |
| (4) | ✓ | ✓ | ✓ | **71.39** ↑ 3.69 | **57.68** ↑ 4.31 |

(b) Ablation on metric learning.

|  | $F_1$ | $mIoU$ |
|---|---|---|
| Proto | 68.17 | 55.23 |
| Triplet Loss | 67.80 | 55.04 |
| Scatter Loss | 69.15 | 56.30 |
| CoSW+Triplet | 71.88 | 58.49 |
| **CoSW+ Scatter** | **72.32** | **59.25** |

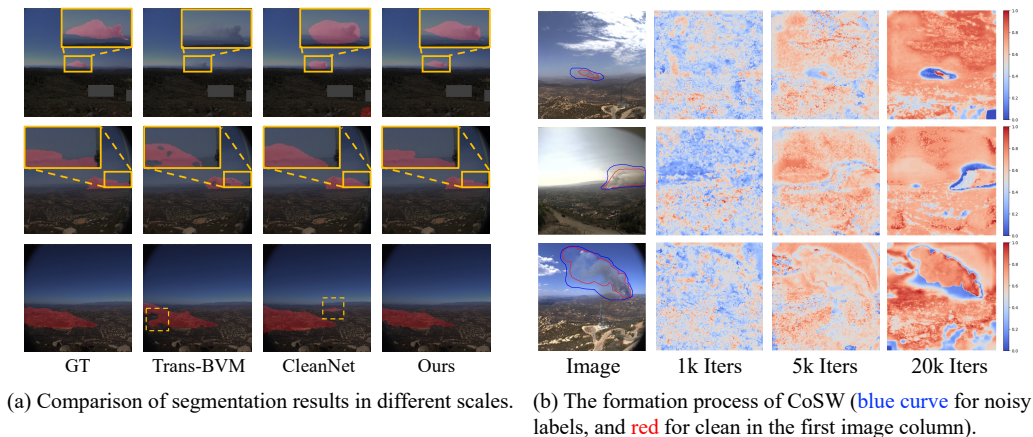

(a) Comparison of segmentation results in different scales. (b) The formation process of CoSW (blue curve for noisy labels, and red for clean in the first image column).

Figure 5: The comparison of segmentation results in different scales and the formation of CoSW.

## 5.2 Comparison on Real-world Label Noise

Tab. 1 and Tab. 2a present the comparative results on the real datasets SmokeSeg and SMOKE5K, respectively. We compare our method with methods from different domains, including general semantic segmentation, SOTA method Trans-BVM [51] for smoke segmentation, segmentation with noisy labels method, SC [52], and the sample weighting method, CleanNet [27]. The results demonstrate that our method achieves the best performance in both datasets. Fig. 5a presents the segmentation results of different methods. The Trans-BVM exhibits high miss detection, while CleanNet is prone to false alarms. In contrast, our method demonstrates the best performance.

## 5.3 Comparison on Synthetic Label Noise

Tab. 2b reports the comparative results of different methods on a synthetic noisy dataset NS-1K. When the dataset is clean or contains only a small amount of low-level noise, Trans-BVM performs the best, as it is specifically designed for smoke detection. However, as the ratio and intensity of noise increase, the performance of the Trans-BVM rapidly deteriorates. In contrast, our method maintains robustness. In the case of maximum noise, our method achieves approximately higher $F_1$ compared to Trans-BVM.

## 5.4 Investigation on CoSW

**Quantitive Ablation.** To investigate the effect of CoSW, we conduct quantitive ablation, as illustrated in Tab. 3a. In the ablation study, the baseline is SegFormer. Tab. 3a(1) replaces the decoder head of SegFormer with prototypes. And there is improvement in model performance, indicating that prototypes inherently provide robustness. In Tab. 3a(2), we incorporate CoSW into the samples, resulting in a significant performance improvement. Tab. 3a(3) focus on utilizing CoSW for prototype updates, as stable updates can also enhance robustness. Finally, in Tab. 3a(4), we combine both approaches, resulting in the best performance achieved.

**How CoSW Formation.** To investigate the reasons behind the effects of CoSW, we visualize the formation process of CoSW, as shown in Fig 5b. It can be observed that as training progresses, CoSW gradually forms its own confidence area. It reduces the weight of the noisy label parts, and even more so for highly transparent areas, as these regions are more prone to noise. This indicates that with the aid of CoSW, the model has developed its own recognition of smoke, rather than being completely governed by labels. This is likely the reason for its robustness against noisy labels. However, the CoSW requires the assumption that the majority of pixels have clean labels. When the label mask is completely noisy, the model is also unable to distinguish the noisy labels, because it can not learn which features are the common of smoke.

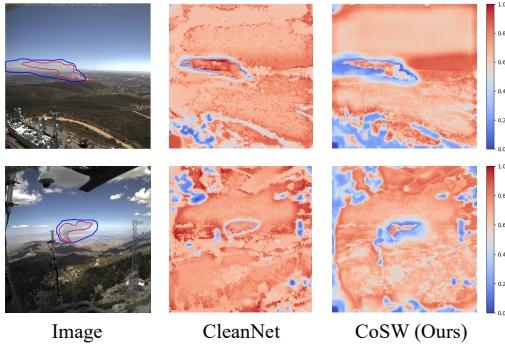

**Qualitative Comparation.** Fig. 6 illustrates the qualitative comparison among prototype-based sample weighting methods, CleanNet, and our CoSW. Although CleanNet can provide a confidence area similar to that of CoSW, its area is less precise. Moreover, the weighting area of CleanNet changes more abruptly and lacks specificity. In contrast, CoSW differentiates between transparent and opaque areas: the weighting in transparent areas is gradual, while it is steeper in transparent areas.

| Image | CleanNet | CoSW (Ours) |

Figure 6: Comparison of sample weighting.

## 5.5 Further Investigation

**Effect of Regularized Scatter Metric Learning.** In Sec. 4.3, we introduce RSML, designed specifically for metric learning under noisy labels. Tab. 3b demonstrates the effect of RSML. It can be observed that using the triplet loss directly in metric learning under noisy labels diminishes the performance of the model. However, incorporating CoSW leads to improved results, and the performance is further enhanced when using the *scatter matrix* as a metric for evaluation.

**Different Entropies.** Our RWE (*cf.* Eq. 5) utilizes Shannon entropy as its foundation in this paper. We have also experimented with different entropies, such as Kapur's entropy and Burg's entropy. The results are shown in Tab. 4. The derivation process is provided in the Appendix B.

Table 4: Different entropies.

|                      | $F_1$ | $mIoU$ |
| -------------------- | ----- | ------ |
| Kapur's Entropy      | 71.34 | 58.38  |
| Burg's Entropy       | 69.20 | 55.72  |
| **Shannon's Entropy** | **72.32** | **59.25** |

## 6 Conclusion

Smoke annotation is prone to noisy labels, which can lead to model instability and result in serious accidents. However, existing methods have not addressed the issue of noisy labels in this field. In this paper, we introduce *conditional sample weighting* (CoSW) to address inconsistent noisy labels in smoke segmentation. CoSW utilizes prototypes as prior information and measures each feature cluster with different criteria to re-weight the samples adaptively. Experimental results show that our CoSW achieves the best performance on both real and synthetic noisy smoke segmentation datasets.

**Limitations.** The CoSW relies on the assumption that the majority of pixels have clean labels. If the degree of noise is very high, the prototype may learn the features of the noise rather than the classes (background and smoke). To determine whether the prototype is clean, it is necessary to introduce clean validation, which is not implemented in our work. This is also a direction for further research.

## Acknowledgements

This work was supported by the National Natural Science Foundation of China (NSFC) under Grant 62173143.

# References

[1] Görkem Algan and Ilkay Ulusoy. Image classification with deep learning in the presence of noisy labels: A survey. *Knowledge-Based Systems*, 215:106771, 2021.

[2] Devansh Arpit, Stanislaw Jastrzebski, Nicolas Ballas, David Krueger, Emmanuel Bengio, Maxinder S Kanwal, Tegan Maharaj, Asja Fischer, Aaron Courville, Yoshua Bengio, et al. A closer look at memorization in deep networks. In *International conference on machine learning*, pages 233–242. PMLR, 2017.

[3] Jane Bromley, Isabelle Guyon, Yann LeCun, Eduard Säckinger, and Roopak Shah. Signature verification using a" siamese" time delay neural network. *Advances in neural information processing systems*, 6, 1993.

[4] Mathilde Caron, Ishan Misra, Julien Mairal, Priya Goyal, Piotr Bojanowski, and Armand Joulin. Unsupervised learning of visual features by contrasting cluster assignments. *Advances in neural information processing systems*, 33:9912–9924, 2020.

[5] Liang-Chieh Chen, Yukun Zhu, George Papandreou, Florian Schroff, and Hartwig Adam. Encoder-decoder with atrous separable convolution for semantic image segmentation. In *Proceedings of the European conference on computer vision (ECCV)*, pages 801–818, 2018.

[6] Thomas Cover and Peter Hart. Nearest neighbor pattern classification. *IEEE transactions on information theory*, 13(1):21–27, 1967.

[7] Marco Cuturi. Sinkhorn distances: Lightspeed computation of optimal transport. *Advances in neural information processing systems*, 26, 2013.

[8] Bo Dong, Pichao Wang, and Fan Wang. Head-free lightweight semantic segmentation with linear transformer. In *Proceedings of the AAAI Conference on Artificial Intelligence*, volume 37, pages 516–524, 2023.

[9] Nanqing Dong and Eric P Xing. Few-shot semantic segmentation with prototype learning. In *BMVC*, volume 3, 2018.

[10] Matthias Dorfer, Rainer Kelz, and Gerhard Widmer. Deep linear discriminant analysis. *arXiv preprint arXiv:1511.04707*, 2015.

[11] Alexander Filonenko, Danilo Cáceres Hernández, and Kang-Hyun Jo. Fast smoke detection for video surveillance using cuda. *IEEE Transactions on Industrial Informatics*, 14(2):725–733, 2017.

[12] Jacob Goldberger, Geoffrey E Hinton, Sam Roweis, and Russ R Salakhutdinov. Neighbourhood components analysis. *Advances in neural information processing systems*, 17, 2004.

[13] Meng-Hao Guo, Cheng-Ze Lu, Qibin Hou, Zhengning Liu, Ming-Ming Cheng, and Shi-Min Hu. Segnext: Rethinking convolutional attention design for semantic segmentation. *Advances in Neural Information Processing Systems*, 35:1140–1156, 2022.

[14] Raia Hadsell, Sumit Chopra, and Yann LeCun. Dimensionality reduction by learning an invariant mapping. In *2006 IEEE computer society conference on computer vision and pattern recognition (CVPR'06)*, volume 2, pages 1735–1742. IEEE, 2006.

[15] Kaiming He, Haoqi Fan, Yuxin Wu, Saining Xie, and Ross Girshick. Momentum contrast for unsupervised visual representation learning. In *Proceedings of the IEEE/CVF conference on computer vision and pattern recognition*, pages 9729–9738, 2020.

[16] Yen-Chia Hsu, Ting-Hao Kenneth Huang, Ting-Yao Hu, Paul Dille, Sean Prendi, Ryan Hoffman, Anastasia Tsuhlares, Jessica Pachuta, Randy Sargent, and Illah Nourbakhsh. Project rise: Recognizing industrial smoke emissions. In *Proceedings of the AAAI Conference on Artificial Intelligence*, volume 35, pages 14813–14821, 2021.

[17] Peter J Huber. *Robust statistics*, volume 523. John Wiley & Sons, 2004.

[18] Edwin T Jaynes. Information theory and statistical mechanics. *Physical review*, 106(4):620, 1957.

[19] Yang Jia, Jie Yuan, Jinjun Wang, Jun Fang, Qixing Zhang, and Yongming Zhang. A saliency-based method for early smoke detection in video sequences. *Fire technology*, 52:1271–1292, 2016.

[20] Lu Jiang, Deyu Meng, Shoou-I Yu, Zhenzhong Lan, Shiguang Shan, and Alexander Hauptmann. Self-paced learning with diversity. *Advances in neural information processing systems*, 27, 2014.

[21] Lu Jiang, Zhengyuan Zhou, Thomas Leung, Li-Jia Li, and Li Fei-Fei. Mentornet: Learning data-driven curriculum for very deep neural networks on corrupted labels. In *International conference on machine learning*, pages 2304–2313. PMLR, 2018.

[22] Tao Jing, Qing-Hao Meng, and Hui-Rang Hou. Smokeseger: A transformer-cnn coupled model for urban scene smoke segmentation. *IEEE Transactions on Industrial Informatics*, 2023.

[23] Tao Jing, Ming Zeng, and Qing-Hao Meng. Smokepose: End-to-end smoke keypoint detection. *IEEE Transactions on Circuits and Systems for Video Technology*, 2023.

[24] Sungyeon Kim, Dongwon Kim, Minsu Cho, and Suha Kwak. Proxy anchor loss for deep metric learning. In *Proceedings of the IEEE/CVF conference on computer vision and pattern recognition*, pages 3238–3247, 2020.

[25] Alexander Kirillov, Eric Mintun, Nikhila Ravi, Hanzi Mao, Chloe Rolland, Laura Gustafson, Tete Xiao, Spencer Whitehead, Alexander C Berg, Wan-Yen Lo, et al. Segment anything. *arXiv preprint arXiv:2304.02643*, 2023.

[26] Teuvo Kohonen. The self-organizing map. *Proceedings of the IEEE*, 78(9):1464–1480, 1990.

[27] Kuang-Huei Lee, Xiaodong He, Lei Zhang, and Linjun Yang. Cleannet: Transfer learning for scalable image classifier training with label noise. In *Proceedings of the IEEE conference on computer vision and pattern recognition*, pages 5447–5456, 2018.

[28] Xiuqing Li, Zhenxue Chen, QM Jonathan Wu, and Chengyun Liu. 3d parallel fully convolutional networks for real-time video wildfire smoke detection. *IEEE Transactions on Circuits and Systems for Video Technology*, 30(1):89–103, 2018.

[29] Or Litany and Daniel Freedman. Soseleto: A unified approach to transfer learning and training with noisy labels. *arXiv preprint arXiv:1805.09622*, 2018.

[30] Chengjiang Long, Jianhui Zhao, Shizhong Han, Lu Xiong, Zhiyong Yuan, Jing Huang, and Weiwei Gao. Transmission: a new feature for computer vision based smoke detection. In *Artificial Intelligence and Computational Intelligence: International Conference, AICI 2010, Sanya, China, October 23-24, 2010, Proceedings, Part I 2*, pages 389–396. Springer, 2010.

[31] Pascal Mettes, Elise Van der Pol, and Cees Snoek. Hyperspherical prototype networks. *Advances in neural information processing systems*, 32, 2019.

[32] Yair Movshovitz-Attias, Alexander Toshev, Thomas K Leung, Sergey Ioffe, and Saurabh Singh. No fuss distance metric learning using proxies. In *Proceedings of the IEEE international conference on computer vision*, pages 360–368, 2017.

[33] Deep Patel and PS Sastry. Adaptive sample selection for robust learning under label noise. In *Proceedings of the IEEE/CVF Winter Conference on Applications of Computer Vision*, pages 3932–3942, 2023.

[34] Qi Qian, Lei Shang, Baigui Sun, Juhua Hu, Hao Li, and Rong Jin. Softtriple loss: Deep metric learning without triplet sampling. In *Proceedings of the IEEE/CVF International Conference on Computer Vision*, pages 6450–6458, 2019.

[35] Florian Schroff, Dmitry Kalenichenko, and James Philbin. Facenet: A unified embedding for face recognition and clustering. In *Proceedings of the IEEE conference on computer vision and pattern recognition*, pages 815–823, 2015.

[36] Jun Shu, Qi Xie, Lixuan Yi, Qian Zhao, Sanping Zhou, Zongben Xu, and Deyu Meng. Meta-weight-net: Learning an explicit mapping for sample weighting. *Advances in neural information processing systems*, 32, 2019.

[37] Jun Shu, Xiang Yuan, Deyu Meng, and Zongben Xu. Cmw-net: Learning a class-aware sample weighting mapping for robust deep learning. *IEEE Transactions on Pattern Analysis and Machine Intelligence*, 2023.

[38] Zeren Sun, Fumin Shen, Dan Huang, Qiong Wang, Xiangbo Shu, Yazhou Yao, and Jinhui Tang. Pnp: Robust learning from noisy labels by probabilistic noise prediction. In *Proceedings of the IEEE/CVF Conference on Computer Vision and Pattern Recognition*, pages 5311–5320, 2022.

[39] Cédric Villani et al. *Optimal transport: old and new*, volume 338. Springer, 2009.

[40] Qiang Wan, Zilong Huang, Jiachen Lu, YU Gang, and Li Zhang. Seaformer: Squeeze-enhanced axial transformer for mobile semantic segmentation. In *The Eleventh International Conference on Learning Representations*, 2022.

[41] Huan Wang, Shuicheng Yan, Dong Xu, Xiaoou Tang, and Thomas Huang. Trace ratio vs. ratio trace for dimensionality reduction. In *2007 IEEE Conference on Computer Vision and Pattern Recognition*, pages 1–8. IEEE, 2007.

[42] Jiang Wang, Yang Song, Thomas Leung, Chuck Rosenberg, Jingbin Wang, James Philbin, Bo Chen, and Ying Wu. Learning fine-grained image similarity with deep ranking. In *Proceedings of the IEEE conference on computer vision and pattern recognition*, pages 1386–1393, 2014.

[43] Kaixin Wang, Jun Hao Liew, Yingtian Zou, Daquan Zhou, and Jiashi Feng. Panet: Few-shot image semantic segmentation with prototype alignment. In *proceedings of the IEEE/CVF international conference on computer vision*, pages 9197–9206, 2019.

[44] Xialli Wang, Aiping Jiang, and Yingli Wang. A segmentation method of smoke in forest-fire image based on fbm and region growing. In *2011 Fourth International Workshop on Chaos-Fractals Theories and Applications*, pages 390–393. IEEE, 2011.

[45] Zhirong Wu, Yuanjun Xiong, Stella X Yu, and Dahua Lin. Unsupervised feature learning via non-parametric instance discrimination. In *Proceedings of the IEEE conference on computer vision and pattern recognition*, pages 3733–3742, 2018.

[46] Qiangqiang Xia, Feifei Lee, and Qiu Chen. Tcc-net: A two-stage training method with contradictory loss and co-teaching based on meta-learning for learning with noisy labels. *Information Sciences*, 639:119008, 2023.

[47] Enze Xie, Wenhai Wang, Zhiding Yu, Anima Anandkumar, Jose M Alvarez, and Ping Luo. Segformer: Simple and efficient design for semantic segmentation with transformers. *Advances in Neural Information Processing Systems*, 34:12077–12090, 2021.

[48] Deng Xing, Yu Zhongming, Wang Lin, and Li Jinlan. Smoke image segmentation based on color model. *Journal on Innovation and Sustainability RISUS*, 6(2):130–138, 2015.

[49] Gao Xu, Yongming Zhang, Qixing Zhang, Gaohua Lin, Zhong Wang, Yang Jia, and Jinjun Wang. Video smoke detection based on deep saliency network. *Fire Safety Journal*, 105:277–285, 2019.

[50] Cheng Xue, Qi Dou, Xueying Shi, Hao Chen, and Pheng-Ann Heng. Robust learning at noisy labeled medical images: Applied to skin lesion classification. In *2019 IEEE 16th International Symposium on Biomedical Imaging (ISBI 2019)*, pages 1280–1283. IEEE, 2019.

[51] Siyuan Yan, Jing Zhang, and Nick Barnes. Transmission-guided bayesian generative model for smoke segmentation. In *Proceedings of the AAAI Conference on Artificial Intelligence*, volume 36, pages 3009–3017, 2022.

[52] Jiachen Yao, Yikai Zhang, Songzhu Zheng, Mayank Goswami, Prateek Prasanna, and Chao Chen. Learning to segment from noisy annotations: A spatial correction approach. *arXiv preprint arXiv:2308.02498*, 2023.

[53] Lujian Yao, Haitao Zhao, Jingchao Peng, Zhongze Wang, and Kaijie Zhao. Fosp: Focus and separation network for early smoke segmentation. In *Proceedings of the AAAI Conference on Artificial Intelligence*, volume 38, pages 6621–6629, 2024.

[54] Feiniu Yuan, Zeshu Dong, Lin Zhang, Xue Xia, and Jinting Shi. Cubic-cross convolutional attention and count prior embedding for smoke segmentation. *Pattern Recognition*, 131:108902, 2022.

[55] Feiniu Yuan, Lin Zhang, Xue Xia, Qinghua Huang, and Xuelong Li. A gated recurrent network with dual classification assistance for smoke semantic segmentation. *IEEE Transactions on Image Processing*, 30:4409–4422, 2021.

[56] Feiniu Yuan, Lin Zhang, Xue Xia, Boyang Wan, Qinghua Huang, and Xuelong Li. Deep smoke segmentation. *Neurocomputing*, 357:248–260, 2019.

[57] Yuhui Yuan, Xilin Chen, and Jingdong Wang. Object-contextual representations for semantic segmentation. In *Computer Vision–ECCV 2020: 16th European Conference, Glasgow, UK, August 23–28, 2020, Proceedings, Part VI 16*, pages 173–190. Springer, 2020.

[58] Chiyuan Zhang, Samy Bengio, Moritz Hardt, Benjamin Recht, and Oriol Vinyals. Understanding deep learning (still) requires rethinking generalization. *Communications of the ACM*, 64(3):107–115, 2021.

[59] Jianmei Zhang, Hongqing Zhu, Pengyu Wang, and Xiaofeng Ling. Att squeeze u-net: A lightweight network for forest fire detection and recognition. *IEEE Access*, 9:10858–10870, 2021.

[60] Minqing Zhang, Jiantao Gao, Zhen Lyu, Weibing Zhao, Qin Wang, Weizhen Ding, Sheng Wang, Zhen Li, and Shuguang Cui. Characterizing label errors: Confident learning for noisy-labeled image segmentation. In *Medical Image Computing and Computer Assisted Intervention–MICCAI 2020: 23rd International Conference, Lima, Peru, October 4–8, 2020, Proceedings, Part I 23*, pages 721–730. Springer, 2020.

[61] Haitao Zhao, Zhihui Lai, Henry Leung, and Xianyi Zhang. Linear discriminant analysis. *Feature Learning and Understanding: Algorithms and Applications*, pages 71–85, 2020.

[62] Tianfei Zhou, Wenguan Wang, Ender Konukoglu, and Luc Van Gool. Rethinking semantic segmentation: A prototype view. In *Proceedings of the IEEE/CVF Conference on Computer Vision and Pattern Recognition*, pages 2582–2593, 2022.

[63] Haidong Zhu, Jialin Shi, and Ji Wu. Pick-and-learn: Automatic quality evaluation for noisy-labeled image segmentation. In *Medical Image Computing and Computer Assisted Intervention–MICCAI 2019: 22nd International Conference, Shenzhen, China, October 13–17, 2019, Proceedings, Part VI 22*, pages 576–584. Springer, 2019.

[64] Yuehua Zhu, Muli Yang, Cheng Deng, and Wei Liu. Fewer is more: A deep graph metric learning perspective using fewer proxies. *Advances in Neural Information Processing Systems*, 33:17792–17803, 2020.

# Appendix

## A  CoSW Derivation

The optimization problem for CoSW is:

$$\max_{P,V} J(P,V) = -\sum_{k=1}^{\Omega K} \frac{N^k}{N} \sum_{n=1}^{N^k} \frac{v_n^k}{N^k} \ln(\frac{v_n^k}{N^k})$$
$$-\gamma \sum_{k=1}^{\Omega K} \frac{N^k}{N} \sum_{n=1}^{N^k} \frac{v_n^k}{N^k} \|\boldsymbol{x}_n^k - \boldsymbol{p}^k\|_2, \tag{14}$$

with the constraint $\sum_{n=1}^{N^k} v_n^k = N^k$, where $N^k$ denotes the number of pixel features belonging to $\omega$. The penalty function $q(\cdot)$ employs the $L_2$ norm. The $\gamma \in [0,1]$ is a *regularization parameter* which controls the degree of punishment.

We can transform the maximization function to incorporate all constraints by Lagrange multipliers:

$$L(P,V) = -\sum_{k=1}^{\Omega K} \frac{N^k}{N} \sum_{n=1}^{N^k} \frac{v_n^k}{N^k} \ln(\frac{v_n^k}{N^k})$$
$$-\gamma \sum_{k=1}^{\Omega K} \frac{N^k}{N} \sum_{n=1}^{N^k} \frac{v_n^k}{N^k} \|\boldsymbol{x}_n^k - \boldsymbol{p}^k\|_2 \tag{15}$$
$$+\sum_{k=1}^{\Omega K} \lambda^k (\sum_{n=1}^{N^k} \frac{v_n^k}{N^k} - 1).$$

Taking the partial derivative of Eq. 15 with respect to $v_n^k$ and $\boldsymbol{p}$ equal to zero, we can obtain the CoSW $v_n^k$ of each sample and the objective values $\hat{\boldsymbol{p}}^k$ for prototype update:

$$\frac{\partial L}{\partial v_n^k} = -\frac{N^k}{N}(\frac{1}{N^k} \ln \frac{v_n^k}{N^k})$$
$$-\gamma \frac{1}{N} \|\boldsymbol{x}_n^k - \boldsymbol{p}^k\|_2 + \lambda^k \frac{1}{N^k} = 0, \tag{16}$$

$$\frac{\partial L}{\partial \boldsymbol{p}^k} = 2\sum_{n=1}^{N^k} \frac{N^k}{N}(\boldsymbol{x}_n^k - \boldsymbol{p}^k)\frac{v_n^k}{N^k} = 0. \tag{17}$$

For Eq. 16, we have $\ln v_n^k/N^k = -\gamma\|\boldsymbol{x}_n^k - \boldsymbol{p}^k\|_2 + \gamma(N/N^k)\lambda^k - 1$. Then

$$\frac{v_n^k}{N^k} = \exp(-\gamma\|\boldsymbol{x}_n^k - \boldsymbol{p}^k\|_2)\exp(\frac{\lambda^k \gamma N}{N^k} - 1). \tag{18}$$

Since $\sum_{n=1}^{N^k} v_n^k = N^k$, we have

$$\exp(\frac{\lambda^k \gamma N}{N^k} - 1) = \frac{1}{\exp(-\gamma\|\boldsymbol{x}_n^k - \boldsymbol{p}^k\|_2)}. \tag{19}$$

Substituting Eq. 19 into Eq. 18, the sample weighting can be obtained:

$$v_n^k = N^k \frac{\exp(-\gamma\|\boldsymbol{x}_n^k - \boldsymbol{p}^k\|_2)}{\sum_{n=1}^{N^k}\exp(-\gamma\|\boldsymbol{x}_n^k - \boldsymbol{p}^k\|_2)}. \tag{20}$$

And for Eq. 17, we have

$$\hat{\boldsymbol{p}}^k = \frac{\sum_{n=1}^{N^k} \boldsymbol{x}_n^k v_n^k}{\sum_{n=1}^{N^k} v_n^k} = \frac{\sum_{n=1}^{N^k} \boldsymbol{x}_n^k \exp(-\gamma\|\boldsymbol{x}_n^k - \boldsymbol{p}^k\|_2)}{\sum_{n=1}^{N^k}\exp(-\gamma\|\boldsymbol{x}_n^k - \boldsymbol{p}^k\|_2)}. \tag{21}$$

# B  Other Entropies

## B.1  Burg's Entropy

For a probability distribution $\Pi = (\pi_1, \pi_2, ..., \pi_N)$ of $N$ random variables $\{x_1, x_2, ..., x_N\}$, Burg's entropy is defined as

$$T^B(\Pi) = -\sum_{i=1}^{N} \ln \pi_i. \tag{22}$$

Based on this, we can define the following entropy:

Total-prototype entropy:

$$T^B = \sum_{k=1}^{\Omega K} \sum_{n=1}^{N^k} \ln v_n^k / N. \tag{23}$$

Within-prototype entropy:

$$T_w^B = \sum_{k=1}^{\Omega K} T_k^B, \tag{24}$$

where $T_k^B = \sum_{n=1}^{N^k} \ln v_n^k / N$ is formulated as the Burg's entropy of the similarities in the n-*th* prototype.

Between-prototype entropy:

$$T_b^B = \sum_{k=1}^{\Omega K} \sum_{n=1}^{N^k} \ln N^k / N. \tag{25}$$

Then, we also have $T^B = T_w^B + T_b^B$. The optimization objective can be formulated as

$$\max_{P,V} J^B(P,V) = -\sum_{k=1}^{\Omega K} \sum_{n=1}^{N^k} \ln\left(\frac{v_n^k}{N^k}\right)$$
$$- \gamma \sum_{k=1}^{\Omega K} \frac{N^k}{N} \sum_{n=1}^{N^k} \frac{v_n^k}{N^k} \|\boldsymbol{x}_n^k - \boldsymbol{p}^k\|_2. \tag{26}$$

We can follow the procedure in Sec. A to solve Eq. 26:

$$v_n^k = N^k \frac{(-\gamma\|\boldsymbol{x}_n^k - \boldsymbol{p}^k\|_2)}{\sum_{n=1}^{N^k}(-\gamma\|\boldsymbol{x}_n^k - \boldsymbol{p}^k\|_2)}, \tag{27}$$

$$\hat{\boldsymbol{p}}^k = \frac{\sum_{n=1}^{N^k} \boldsymbol{x}_n^k (-\gamma\|\boldsymbol{x}_n^k - \boldsymbol{p}^k\|_2)}{\sum_{n=1}^{N^k}(-\gamma\|\boldsymbol{x}_n^k - \boldsymbol{p}^k\|_2)}, \tag{28}$$

## B.2  Kapur's Entropy

For a probability distribution $\Pi = (\pi_1, \pi_2, ..., \pi_N)$ of $N$ random variables $\{x_1, x_2, ..., x_N\}$, Kapur's entropy is defined as

$$T^K(\Pi) = -\sum_{i=1}^{N} \pi_i \ln \pi_i - \sum_{i=1}^{N}(1 - \pi_i)\ln(1 - \pi_i). \tag{29}$$

Based on this, we can define within-prototype Kapur's entropy as

$$T_w = -\sum_{k=1}^{\Omega K} \frac{N^k}{N} \sum_{n=1}^{N^k} \frac{v_n^k}{N^k} \ln\left(\frac{v_n^k}{N^k}\right)$$
$$- \sum_{k=1}^{\Omega K} \frac{N^k}{N} \sum_{n=1}^{N^k}\left(1 - \frac{v_n^k}{N^k}\right)\ln\left(1 - \frac{v_n^k}{N^k}\right). \tag{30}$$

For Shannon's entropy and Burg's entropy, we have $T = T_w + T_b$ and $T^B = T_w^B + T_b^B$. However, in Kapur's entropy, it is not the case. For a given classification problem, maximizing the within-prototype Kapur's entropy is different from maximizing Kapur's entropy on the whole dataset. This means we cannot simply consider maximizing the entropy of each prototype independently. But for consistency and comparison, we also use the within-class Kapur's entropy instead of the within-class entropy to design the objective function.

Based on Eq. 30, we have the following objective function:

$$
\begin{aligned}
\max_{P,V} J^K(P, V) = &-\sum_{k=1}^{\Omega K} \frac{N^k}{N} \sum_{n=1}^{N^k} \frac{v_n^k}{N^k} \ln\left(\frac{v_n^k}{N^k}\right) \\
&- \sum_{k=1}^{\Omega K} \frac{N^k}{N} \sum_{n=1}^{N^k} (1 - \frac{v_n^k}{N^k}) \ln(1 - \frac{v_n^k}{N^k}) \\
&- \gamma \sum_{k=1}^{\Omega K} \frac{N^k}{N} \sum_{n=1}^{N^k} \frac{v_n^k}{N^k} \|\boldsymbol{x}_n^k - \boldsymbol{p}^k\|_2 .
\end{aligned}
\tag{31}
$$

The likelihood value and the objective of the prototype update can be obtained by following the procedure in Sec. A:

$$
v_n^k = N^k \frac{1}{1 + \exp(-\|\boldsymbol{x}_n^k - \boldsymbol{p}^k\|_2 - \lambda_k)^\gamma},
\tag{32}
$$

$$
\hat{\boldsymbol{p}}^k = \sum_{n=1}^{N^k} \frac{\boldsymbol{x}_n^k}{1 + \exp(-\|\boldsymbol{x}_n^k - \boldsymbol{p}^k\|_2 - \lambda_k)^\gamma},
\tag{33}
$$

where $\lambda_k$ are the solutions of $\sum_{n=1}^{N^k} v_n^k = N^k$.

## C  Details of Pixel-Prototype Matching (P2M)

### C.1  Derivation of P2M

The P2M is based on the matching process in ProtoSeg [62] and SwAV [4]. The optimization problem to be solved is

$$
\min_{\boldsymbol{M}^\omega} \text{Tr}(\boldsymbol{M}^{\omega\top} \boldsymbol{C}^\omega),
$$
$$
s.t. \quad \boldsymbol{M}^\omega \in \{0,1\}^{K \times N^\omega}, \ \boldsymbol{M}^{\omega\top} \mathbf{1}^K = \mathbf{1}^{N^\omega}, \boldsymbol{M}^\omega \mathbf{1}^{N^\omega} = \frac{N^\omega}{K} \mathbf{1}^K,
\tag{34}
$$

where $\boldsymbol{M}^\omega \in \mathbb{R}^{K \times N^\omega}$ is the matching strategy and $\boldsymbol{C}^\omega \in \mathbb{R}^{K \times N^\omega}$ represents the cost matrix that measures the distance between pixel features and prototypes:

$$
\boldsymbol{C}_{ij}^\omega = \|\boldsymbol{x}_j^\omega - \boldsymbol{p}_i^\omega\|_2, i = 1, 2, \cdots, K, j = 1, 2, \cdots, N^\omega.
\tag{35}
$$

Eq. 34 is a typical transportation problem [39] and solving it directly using linear programming is time-consuming. One fast approach involves incorporating an *entropy regularization* term $\tau H(\boldsymbol{M}^\omega)$ to facilitate the relaxation of $\boldsymbol{M}^\omega$ and acquire an approximate solution [7]:

$$
\min_{\boldsymbol{M}^\omega} \text{Tr}(\boldsymbol{M}^{\omega\top} \boldsymbol{C}^\omega) - \tau H(\boldsymbol{M}^\omega),
$$
$$
s.t. \quad \boldsymbol{M}^\omega \in \mathbb{R}_+^{K \times N^\omega}, \ \boldsymbol{M}^{\omega\top} \mathbf{1}^K = \mathbf{1}^{N^\omega}, \boldsymbol{M}^\omega \mathbf{1}^{N^\omega} = \frac{N^\omega}{K} \mathbf{1}^K,
\tag{36}
$$

where $H(\boldsymbol{M}^\omega) = -\sum_{ij} \boldsymbol{M}_{ij}^\omega \log \boldsymbol{M}_{ij}^\omega$ and the $\tau > 0$ is applied to control the smoothness of the matching strategy. We set $\tau = 0.05$. The solution of Eq. 36 can be given by:

$$
\boldsymbol{M}^\omega = \text{Diag}(\boldsymbol{u}) \exp\left(\frac{\boldsymbol{C}^\omega}{-\tau}\right) \text{Diag}(\boldsymbol{v}),
\tag{37}
$$

where $\boldsymbol{u} \in \mathbb{R}^K$ and $\boldsymbol{v} \in \mathbb{R}^{N^\omega}$ represent renormalization vectors. They can be calculated by the iterative Sinkhorn-Knopp algorithm [7] which is efficient on GPU as it only relies on matrix multiplications.

Table 5: Impact of the number of iterations in Sinkhorn-Knopp algorithm (dataset: SmokeSeg, backbone: MiT-B3).

| # Sinkhorn Iterations | 1 | 3 | 5 | 10 | 20 |
|:---:|:---:|:---:|:---:|:---:|:---:|
| $F_1$ | 66.40 | **72.32** | 72.22 | 72.15 | 72.10 |
| $mIoU$ | 52.78 | **59.83** | 59.63 | 59.38 | 59.23 |

## C.2 The Number of Sinkhorn Iterations

We also investigate the impact of the number of normalization steps on model performance, as shown in Tab. 5. It can be observed that when iteration=1, the model performance is poor. We think this is due to an insufficient number of steps, which fails to adequately match the pixels and prototypes. As the number of iterations increases, the performance of the model improves and tends to stabilize.

## C.3 The Number of Prototypes Per Class

Tab. 6 reports the performance of different numbers of prototypes. It can be observed that as the number of prototypes increases from 1 to 3, there is a significant improvement. This demonstrates that the polycentric embedding space formed by multiple prototypes indeed enhances the representation of the model. The growth becomes slow when the number exceeds 10.

Table 6: Impact of the number of prototypes (dataset: SmokeSeg, backbone: MiT-B3).

| # Prototype Number | 1 | 3 | 5 | 10 | 20 |
|:---:|:---:|:---:|:---:|:---:|:---:|
| $F_1$ | 68.73 | 70.85 | 71.71 | 72.32 | 72.43 |
| $mIoU$ | 56.50 | 58.39 | 59.23 | 59.83 | 59.92 |

